# ICEG Morphology Classification using an Analogue VLSI Neural Network

**Richard Coggins, Marwan Jabri, Barry Flower and Stephen Pickard**
Systems Engineering and Design Automation Laboratory
Department of Electrical Engineering J03,
University of Sydney, 2006, Australia.
Email: richardc@sedal.su.oz.au

## Abstract

An analogue VLSI neural network has been designed and tested to perform cardiac morphology classification tasks. Analogue techniques were chosen to meet the strict power and area requirements of an Implantable Cardioverter Defibrillator (ICD) system. The robustness of the neural network architecture reduces the impact of noise, drift and offsets inherent in analogue approaches. The network is a 10:6:3 multi-layer perceptron with on chip digital weight storage, a bucket brigade input to feed the Intracardiac Electrogram (ICEG) to the network and has a winner take all circuit at the output. The network was trained in loop and included a commercial ICD in the signal processing path. The system has successfully distinguished arrhythmia for different patients with better than 90% true positive and true negative detections for dangerous rhythms which cannot be detected by present ICDs. The chip was implemented in $1.2um$ CMOS and consumes less than 200nW maximum average power in an area of $2.2 \times 2.2mm^2$.

## 1 INTRODUCTION

To the present time, most ICDs have used timing information from ventricular leads only to classify rhythms which has meant some dangerous rhythms can not be distinguished from safe ones, limiting the use of the device. Even two lead

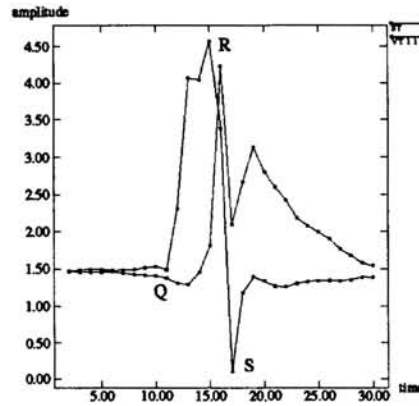

Figure 1: The Morphology of ST and VT retrograde 1:1.

atrial/ventricular systems fail to distinguish some rhythms when timing information alone is used [Leong and Jabri, 1992]. A case in point is the separation of Sinus Tachycardia (ST) from Ventricular Tachycardia with 1:1 retrograde conduction. ST is a safe arrhythmia which may occur during vigorous exercise and is characterised by a heart rate of approximately 120 beats/minute. VT retrograde 1:1 also occurs at the same low rate but can be a potentially fatal condition. False negative detections can cause serious heart muscle injury while false positive detections deplete the batteries, cause patient suffering and may lead to costly transplantation of the device. Figure 1 shows however, the way in which the morphology changes on the ventricular lead for these rhythms. Note, that the morphology change is predominantly in the "QRS complex" where the letters QRS are the conventional labels for the different points in the conduction cycle during which the heart is actually pumping blood.

For a number of years, researchers have studied template matching schemes in order to try and detect such morphology changes. However, techniques such as correlation waveform analysis [Lin et. al., 1988], though quite successful are too computationally intensive to meet power requirements. In this paper, we demonstrate that an analogue VLSI neural network can detect such morphology changes while still meeting the strict power and area requirements of an implantable system. The advantages of an analogue approach are born out when one considers that an energy efficient analogue to digital converter such as [ Kusumoto et. al., 1993] uses 1.5nJ per conversion implying 375nW power consumption for analogue to digital conversion of the ICEG alone. Hence, the integration of a bucket brigade device and analogue neural network provides a very efficient way of interfacing to the analogue domain. Further, since the network is trained in loop with the ICD in real time, the effects of device offsets, noise, QRS detection jitter and signal distortion in the analogue circuits are largely alleviated.

The next section discusses the chip circuit designs. Section 3 describes the method

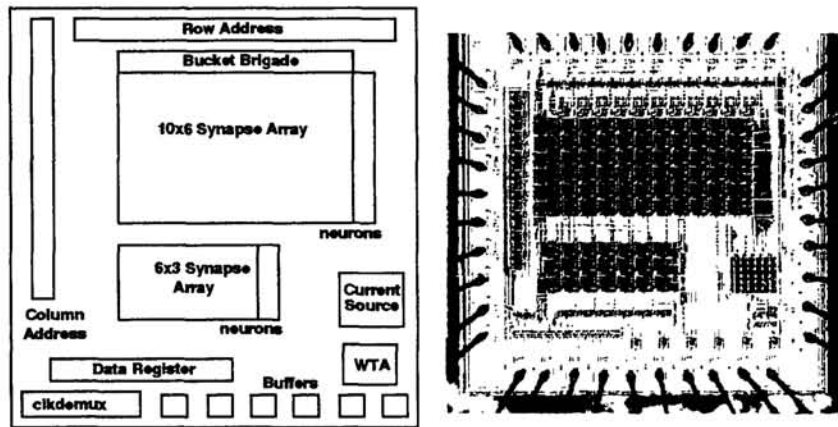

Figure 2: Floor Plan and Photomicrograph of the chip

used to train the network for the morphology classification task. Section 4 describes the classifier performance on seven patients with arrhythmia which can not be distinguished using the heart rate only. Section 5 summarises the results, remaining problems and future directions for the work.

## 2 ARCHITECTURE

The neural network chip consists of a 10:6:3 multilayer perceptron, an input bucket brigade device (BBD) and a winner take all (WTA) circuit at the output. A floor plan and photomicrograph of the chip appears in figure 2. The BBD samples the incoming ICEG at a rate of 250Hz. For three class problems, the winner take all circuit converts the winning class to a digital signal. For the two class problem considered in this paper, a simple thresholding function suffices. The following subsections briefly describe the functional elements of the chip. The circuit diagrams for the chip building blocks appear in figure 3.

### 2.1 BUCKET BRIGADE DEVICE

One stage of the bucket brigade circuit is shown in figure 3. The BBD uses a two phase clock to shift charge from cell to cell and is based on a design by Leong [Leong, 1992]. The BBD operates by transferring charge deficits from S to D in each of the cells. PHI1 and PHI2 are two phase non-overlapping clocks. The cell is buffered from the synapse array to maintain high charge transfer efficiency. A sample and hold facility is provided to store the input on the gates of the synapses. The BBD clocks are generated off chip and are controlled by the QRS complex detector in the ICD.

### 2.2 SYNAPSE

This synapse has been used on a number of neural network chips previously. e.g. [Coggins et. al., 1994]. The synapse has five bits plus sign weight storage which

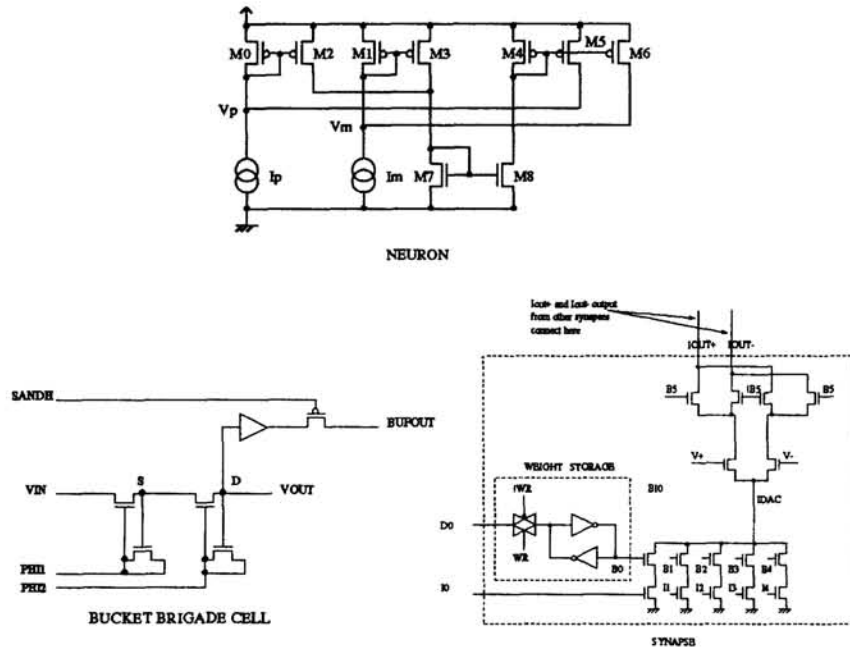

Figure 3: Neuron, Bucket Brigade and Synapse Circuit Diagrams.

sets the bias to a differential pair which performs the multiplication. The bias references for the weights are derived from a weighted current source in the corner of the chip. A four quadrant multiplication is achieved by the four switches at the top of the differential pair.

## 2.3   NEURON

Due to the low power requirements, the bias currents of the synapse arrays are of the order of hundreds of nano amps, hence the neurons must provide an effective resistance of many mega ohms to feed the next synapse layer while also providing gain control. Without special high resistance polysilicon, simple resistive neurons use prohibitive area. However, for larger networks with fan-in much greater than ten, an additional problem of common mode cancellation is encountered. That is, as the fan-in increases, a larger common mode range is required or a cancellation scheme using common mode feedback is needed.

The neuron of figure 3 implements such a cancellation scheme. The mirrors M0/M2 and M1/M3 divide the input current and facilitate the sum at the drain of M7. M7/M8 mirrors the sum so that it may be split into two equal currents by the mirrors formed by M4, M5 and M6 which are then subtracted from the input currents. Thus, the differential voltage $V_p - V_m$ is a function of the transistor transconductances, the common mode input current and the feedback factor. The gain of the neuron can be controlled by varying the width to length ratio of the mirror transistors M0 and M1. The implementation in this case allows seven gain combinations, using a three bit RAM cell to store the gain.

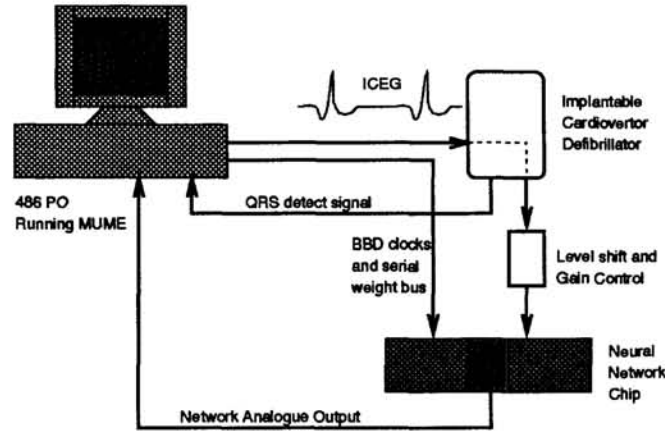

Figure 4: Block Diagram of the Training and Testing System.

The importance of a common mode cancellation scheme for large networks can be seen when compared to the straight forward approach of resistive or switched capacitor neurons. This may be illustrated by considering the energy usage of the two approaches. Firstly, we need to define the required gain of the neuron as a function of its fan-in. If we assume that useful inputs to the network are mostly sparse, i.e. with a small fraction of non-zero values, then the gain is largely independent of the fan-in, yet the common mode signal increases linearly with fan-in. For the case of a neuron which does not cancel the common mode, the power supply voltage must be increased to accommodate the common mode signal, thus leading to a quadratic increase in energy use with fan-in. A common mode cancelling neuron on the other hand, suffers only a linear increase in energy use with fan-in since extra voltage range is not required and the increased energy use arises only due to the linear increase in common mode current.

## 3   TRAINING SYSTEM

The system used to train and test the neural network is shown in figure 4. Control of training and testing takes place on the PC. The PC uses a PC-LAB card to provide analogue and digital I/O. The PC plays the ICEG signal to the input of the commercial ICD in real time. Note, that the PC is only required for initially training the network and in this case as a source of the heart signal. The commercial ICD performs the function of QRS complex detection using analogue circuits. The QRS complex detection signal is then used to freeze the BBD clocks of the chip, so that a classification can take place.

When training, a number of examples of the arrhythmia to be classified are selected from a single patient data base recorded during an electrophysiological study and previously classified by a cardiologist. Since most of the morphological information is in the QRS complex, only these segments of the data are repeatedly presented to

| Patient | % Training Attempts Converged | | | | Average Iterations |
|---|---|---|---|---|---|
| | Run 1 | | Run 2 | | |
| | H = 3 | H = 6 | H = 3 | H = 6 | |
| 1 | 80 | 10 | 60 | 60 | 62 |
| 2 | 80 | 100 | 0 | 10 | 86 |
| 3 | 0 | 0 | 0 | 10 | 101 |
| 4 | 60 | 10 | 40 | 40 | 77 |
| 5 | 100 | 80 | 0 | 60 | 44 |
| 6 | 100 | 40 | 60 | 60 | 46 |
| 7 | 80 | 100 | 40 | 100 | 17 |

Table 1: Training Performance of the system on seven patients.

the network. The weights are adjusted according to the training algorithm running on the PC using the analogue outputs of the network to reduce the output error. The PC writes weights to the chip via the digital I/Os of the PC-LAB card and the serial weight bus of network. The software package implementing the training and testing, called MUME [Jabri et. al., 1992], provides a suite of training algorithms and control options. Online training was used due to its success in training small networks and because the presentation of the QRS complexes to the network was the slowest part of the training procedure. The algorithm used for weight updates in this paper was summed weight node perturbation [Flower and Jabri, 1993].

The system was trained on seven different patients separately all of whom had VT with 1: 1 retrograde conduction. Note, that patient independent training has been tried but with mixed results [Tinker, 1992]. Table 1 summarises the training statistics for the seven patients. For each patient and each architecture, five training runs were performed starting from a different random initial weight set. Each of the patients was trained with eight of each class of arrhythmia. The network architecture used was 10:H:1, where H is the number of hidden layer neurons and the unused neurons being disabled by setting their input weights to zero. Two sets of data were collected denoted Run 1 and Run 2. Run 1 corresponded to output target values of $\pm 0.6V$ within margin $0.45V$ and Run 2 to output target values of $\pm 0.2V$ within margin $0.05V$. A training attempt was considered to have converged when the training set was correctly classified within two hundred training iterations. Once the morphologies to be distinguished have been learned for a given patient, the remainder of the patient data base is played back in a continuous stream and the outputs of the classifier at each QRS complex are logged and may be compared to the classifications of a cardiologist. The resulting generalisation performance is discussed in the next section.

## 4   MORPHOLOGY CLASSIFIER GENERALISATION PERFORMANCE

Table 2 summarises the generalisation performance of the system on the seven patients for the training attempts which converged. Most of the patients show a correct classification rate better than 90% for at least one architecture on one of the

| Patient | No. of Complexes | | % Correct Classifications Run 1 | | | |
|---|---|---|---|---|---|---|
| | | | H = 3 | | H = 6 | |
| | ST | VT | ST | VT | ST | VT |
| 1 | 440 | 61 | 89±10 | 89±3 | 58±0 | 99±0 |
| 2 | 94 | 57 | 99±1 | 99±1 | 100±0 | 99±1 |
| 3 | 67 | 146 | - | - | - | - |
| 4 | 166 | 65 | 66±44 | 76±37 | 99±1 | 50±3 |
| 5 | 61 | 96 | 82±1 | 75±13 | 94±6 | 89±9 |
| 6 | 61 | 99 | 84±8 | 97±1 | 90±5 | 99±1 |
| 7 | 28 | 80 | 98±5 | 97±3 | 99±1 | 99±1 |
| | | | % Correct Classifications Run 2 | | | |
| 1 | 440 | 61 | 88±2 | 99±1 | 86±14 | 99±1 |
| 2 | 94 | 57 | - | - | 94±6 | 94±3 |
| 3 | 67 | 146 | 84±2 | 99±1 | - | - |
| 4 | 166 | 65 | 76±18 | 59±2 | 87±7 | 100±0 |
| 5 | 61 | 96 | 88±2 | 49±5 | 84±1 | 82±5 |
| 6 | 61 | 99 | 92±6 | 90±10 | 99±1 | 99±1 |
| 7 | 28 | 80 | 94±3 | 99±0 | 94±3 | 92±3 |

Table 2: Generalisation Performance of the system on seven patients.

runs, whereas, a timing based classifier can not separate these arrhythmia at all. For each convergent weight set the network classified the test set five times. Thus, the "% Correct" columns denote the mean and standard deviation of the classifier performance with respect to both training and testing variations. By duty cycling the bias to the network and buffers, the chip dissipates less than 200nW power for a nominal heart rate of 120 beats/minute during generalisation.

## 5 DISCUSSION

Referring to table 1 we see that the patient 3 data was relatively difficult to train. However, for the one occasion when training converged generalisation performance was quite acceptable. Inspection of this patients data showed that typically, the morphologies of the two rhythms were very similar. The choice of output targets, margins and architecture appear to be patient dependent and possibly interacting factors. Although larger margins make training easier for some patients they appear to also introduce more variability in generalisation performance. This may be due to the non-linearity of the neuron circuit. Further experiments are required to optimise the architecture for a given patient and to clarify the effect of varying targets, margins and neuron gain. Penalty terms could also be added to the error function to minimise the possibility of missed detections of the dangerous rhythm.

The relatively slow rate of the heart results in the best power consumption being obtained by duty cycling the bias currents to the synapses and the buffers. Hence, the bias settling time of the weighted current source is the limiting factor for reducing power consumption further for this design. By modifying the connection of the current source to the synapses using a bypassing technique to reduce transients in

the weighted currents, still lower power consumption could be achieved.

# 6    CONCLUSION

The successful classification of a difficult cardiac arrhythmia problem has been demonstrated using an analogue VLSI neural network approach. Furthermore, the chip developed has shown very low power consumption of less than 200nW, meeting the requirements of an implantable system. The chip has performed well, with over 90% classification performance for most patients studied and has proved to be robust when the real world influence of analogue QRS detection jitter is introduced by a commercial implantable cardioverter defibrillator placed in the signal path to the classifier.

### Acknowledgements

The authors acknowledge the funding for the work in this paper provided under Australian Generic Technology Grant Agreement No. 16029 and thank Dr. Phillip Leong of the University of Sydney and Dr. Peter Nickolls of Telectronics Pacing Systems Ltd., Australia for their helpful suggestions and advice.

### References

[Castro et. al., 1993] H.A. Castro, S.M. Tam, M.A. Holler, "Implementation and Performance of an analogue Nonvolatile Neural Network," *Analogue Integrated Circuits and Signal Processing*, vol. 4(2), pp. 97-113, September 1993.

[Lin et. al., 1988] D. Lin, L.A. Dicarlo, and J.M. Jenkins, "Identification of Ventricular Tachycardia using Intracavitary Electrograms: analysis of time and frequency domain patterns," *Pacing & Clinical Electrophysiology*, pp. 1592–1606, November 1988.

[Leong, 1992] P.H.W. Leong, Arrhythmia Classification Using Low Power VLSI, PhD Thesis, University of Sydney, Appendix B, 1992.

[ Kusumoto et. al., 1993] K. Kusumoto et. al., "A 10bit 20Mhz 30mW Pipelined Interpolating ADC," *ISSCC, Digest of Technical Papers*, pp. 62-63, 1993.

[Leong and Jabri, 1992] P.H.W. Leong and M. Jabri, "MATIC - An Intracardiac Tachycardia Classification System", *Pacing & Clinical Electrophysiology*, September 1992.

[Coggins et. al., 1994] R.J. Coggins and M.A. Jabri, "WATTLE: A Trainable Gain Analogue VLSI Neural Network", *NIPS6*, Morgan Kauffmann Publishers, 1994.

[Jabri et. al., 1992] M.A. Jabri, E.A. Tinker and L. Leerink, "MUME- A Multi-Net-Multi-Architecture Neural Simulation Environment", Neural Network Simulation Environments, Kluwer Academic Publications, January, 1994.

[Flower and Jabri, 1993] B. Flower and M. Jabri, "Summed Weight Neuron Perturbation: an O(N) improvement over Weight Perturbation," *NIPS5*, Morgan Kauffmann Publishers, pp. 212-219, 1993.

[Tinker, 1992] E.A. Tinker, "The SPASM Algorithm for Ventricular Lead Timing and Morphology Classification," SEDAL ICEG-RPT-016-92, Department of Electrical Engineering, University of Sydney, 1992.